# Place Cells and Spatial Navigation based on 2d Visual Feature Extraction, Path Integration, and Reinforcement Learning

A. Arleo*    F. Smeraldi    S. Hug    W. Gerstner

Centre for Neuro-Mimetic Systems, MANTRA
Swiss Federal Institute of Technology Lausanne,
CH-1015 Lausanne EPFL, Switzerland

## Abstract

We model hippocampal place cells and head-direction cells by combining allothetic (visual) and idiothetic (proprioceptive) stimuli. Visual input, provided by a video camera on a miniature robot, is preprocessed by a set of Gabor filters on 31 nodes of a log-polar retinotopic graph. Unsupervised Hebbian learning is employed to incrementally build a population of localized overlapping place fields. Place cells serve as basis functions for reinforcement learning. Experimental results for goal-oriented navigation of a mobile robot are presented.

## 1 Introduction

In order to achieve spatial learning, both animals and artificial agents need to autonomously locate themselves based on available sensory information. Neurophysiological findings suggest the spatial self-localization of rodents is supported by place-sensitive and direction-sensitive cells. *Place cells* in the rat Hippocampus provide a spatial representation in allocentric coordinates [1]. A place cell exhibits a high firing rate only when the animal is in a specific region of the environment, which defines the *place field* of the cell. *Head-direction cells* observed in the hippocampal formation encode the animal's allocentric heading in the azimuthal plane [2]. A directional cell fires maximally only when the animal's heading is equal to the cell's *preferred direction*, regardless of the orientation of the head relative to the body, of the rat's location, or of the animal's behavior.

We ask two questions. *(i)* How do we get place fields from visual input [3]? This question is non-trivial given that visual input depends on the direction of gaze. We present a computational model which is consistent with several neurophysiological findings concerning biological head-direction and place cells. Place-coding and directional sense are provided by two coupled neural systems, which interact with each other to form a single substrate for spatial navigation (Fig. 1(a)). Both systems rely on allothetic cues (e.g., visual stimuli) as well as idiothetic signals (e.g., proprioceptive cues) to establish stable internal representations. The resulting representation consists of overlapping place fields with properties similar to those of hippocampal place cells. *(ii)* What's the use of place cells for navigation

---

*Corresponding author, *angelo.arleo@epfl.ch*

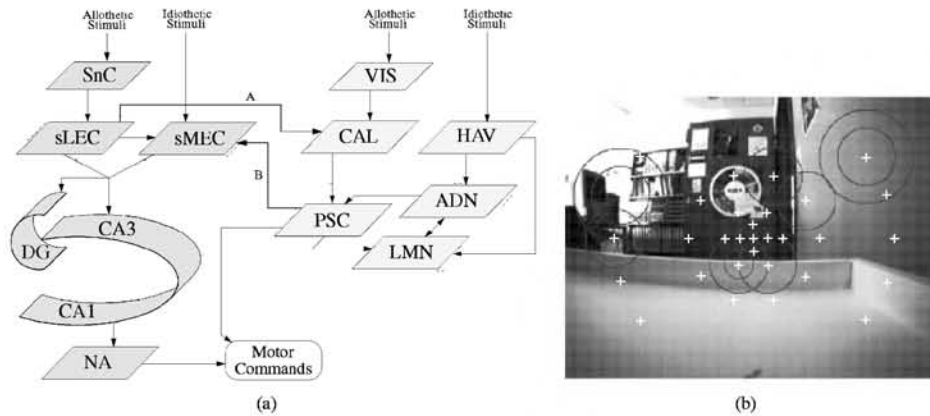

Figure 1: (a) An overview of the entire system. Dark grey areas are involved in space representation, whereas light grey components form the head-direction circuit. Glossary: SnC: hypothetical snapshot cells, sLEC: superficial lateral entorhinal cortex, sMEC: superficial medial entorhinal cortex, DG: dentate gyrus, CA3-CA1: hippocampus proper, NA: nucleus accumbens, VIS: visual bearing cells, CAL: hypothetical calibration cells, HAV: head angular velocity cells, PSC: postsubiculum, ADN: anterodorsal nucleus, LMN: lateral mammillary nuclei. (b) A visual scene acquired by the robot during spatial learning. The image resolution is $422 \times 316$ pixels. The retinotopic sampling grid (white crosses) is employed to sample visual data by means of Gabor decomposition. Black circles represent maximally responding Gabor filters (the circle radius varies as a function of the filter's spatial frequency).

[1]? We show that a representation by overlapping place fields is a natural "state space" for reinforcement learning. A direct implementation of reinforcement learning on real visual streams would be impossible given the high dimensionality of the visual input space. A place field representation extracts the low-dimensional view manifold on which efficient reinforcement learning is possible.

To validate our model in real task-environment contexts, we have tested it on a Khepera miniature mobile robot. Visual information is supplied by an on-board video camera. Eight infrared sensors provide obstacle detection and measure ambient light. Idiothetic signals are provided by the robot's dead-reckoning system. The experimental setup consists of an open-field square arena of about $80 \times 80$ cm in a standard laboratory background (Fig. 1(b)).

The vision-based localization problem consists of *(i)* detecting a convenient low-dimensional representation of the continuous high-dimensional input space (images have a resolution of $422 \times 316$ pixels), *(ii)* learning the mapping function from the visual sensory space to points belonging to this representation. Since our robot moves on a two-dimensional space with a camera pointing in the direction of motion, the high-dimensional visual space is not uniformly filled. Rather, all input data points lie on a low-dimensional surface which is embedded in an Euclidean space whose dimensionality is given by the total number of camera pixels. This low-dimensional description of the visual space is referred to as *view manifold* [4].

## 2 Extracting the low-dimensional view manifold

Hippocampal place fields are determined by a combination of highly-processed multimodal sensory stimuli (e.g., visual, auditory, olfactory, and somatosensory cues) whose mutual relationships code for the animal's location [1]. Nevertheless, experiments on rodents suggest that vision plays an eminent role in determining place cell activity [5]. Here, we focus on

the visual pathway, and we propose a processing in four steps.

As a first step, we place a retinotopic sampling grid on the image (Fig. 1(b)). In total we have 31 grid points with high resolution only in a localized region of the view field (*fovea*), whereas peripheral areas are characterized by a low-resolution vision [6]. At each point of the grid we place 24 Gabor filters with different orientations and amplitudes. Gabor filters [7] provide a suitable mathematical model for biological simple cells [8]. Specifically, we employ a set of *modified Gabor filters* [9]. A modified Gabor filter $f_i$, tuned to orientation $\phi_j$ and angular frequency $\omega_l = e^{\xi_l}$, corresponds to a Gaussian in the *Log-polar* frequency plane rather than in the frequency domain itself, and is defined by the Fourier function

$$\hat{G}(\xi, \phi) = A \cdot e^{-(\xi-\xi_j)^2/2\sigma_\xi^2} \cdot e^{-(\phi-\phi_l)^2/2\sigma_\phi^2} \tag{1}$$

where $A$ is a normalization term, and $(\xi, \phi)$ are coordinates in the Log-polar Fourier plane

$$(\xi, \phi) = \big(\log\|(\omega_x, \omega_y)\|, \ \arctan(\omega_y/\omega_x)\big) \tag{2}$$

A key property of the Log-polar reference frame is that translations along $\phi$ correspond to rotations in the image domain, while translations along $\xi$ correspond to scaling the image. In our implementation, we build a set of 24 modified Gabor filters, $\mathcal{F} = \{f_i(\omega_l, \phi_j) \mid 1 \leq l \leq 3, 1 \leq j \leq 8\}$, obtained by taking 3 angular frequencies $\omega_1$, $\omega_2$, $\omega_3$, and 8 orientations $\phi_1, \ldots, \phi_8$.

As a second step, we take the *magnitude* of the responses of these Gabor filters for detecting visual properties within video streams. While the Gabor filter itself has properties related to simple cells, the amplitude of the complex response does not depend on the exact position within the receptive field and has therefore properties similar to cortical complex cells. Thus, given an image $I(x, y)$, we compute the magnitude of the response of all $f_i$ filters for each retinal point $\vec{g}$

$$r_i(\vec{g}) = \left(\left(\sum_{\vec{x}} Re(f_i(\vec{x})) \cdot I(\vec{g}+\vec{x})\right)^2 + \left(\sum_{\vec{x}} Im(f_i(\vec{x})) \cdot I(\vec{g}+\vec{x})\right)^2\right)^{\frac{1}{2}} \tag{3}$$

where $\vec{x}$ varies over the area occupied by the filter $f_i$ in the spatial domain.

The third step within the visual pathway of our model, consists of interpreting visual cues by means of neural activity. We take a population of hypothetical snapshot cells (SnC in Fig. 1(a)) one synapse downstream from the Gabor filter layer. Let $k$ be an index over all $K$ filters forming the retinotopic grid. Given a new image $I$, a snapshot cell $s \in$ SnC is created which receives afferents from all $f_k$ filters. Connections from filters $f_k$ to cell $s$ are initialized according to $w_{sk} = r_k, \forall k \in K$. If, at a later point, the robot sees an image $I'$, the firing activity $r_s$ of cell $s \in$ SnC is computed by

$$r_s = e^{-(\frac{1}{K}\sum_k |r_k - w_{sk}|)^2/2\sigma^2} \tag{4}$$

where $r_k$ are the Gabor filter responses to image $I'$. Eq. 4 defines a radial basis function in the filter space that measures the similarity of the current image to the image stored in the weights $w_{sk}$. The width $\sigma$ determines the discrimination capacity of the system for visual scene recognition.

As final step, we apply unsupervised Hebbian learning to achieve spatial coding one synapse downstream from the SnC layer (sLEC in Fig. 1(a)). Indeed, the snapshot cell activity $r_s$ defined in Eq. 4 depends on the robot's gaze direction, and does not code for a spatial location. In order to collect information from several gaze directions, the robot takes four snapshots corresponding to north, east, south, and west views at each location visited during exploration. To do this, it relies on the allocentric compass information provided by the directional system [2, 10]. For each visited location the robot creates four SnC snapshot cells, which are bound together to form a place cell in the sLEC layer. Thus, sLEC cell activity depends on a combination of several visual cues, which results in non-directional place fields (Fig. 2(a)) [11].

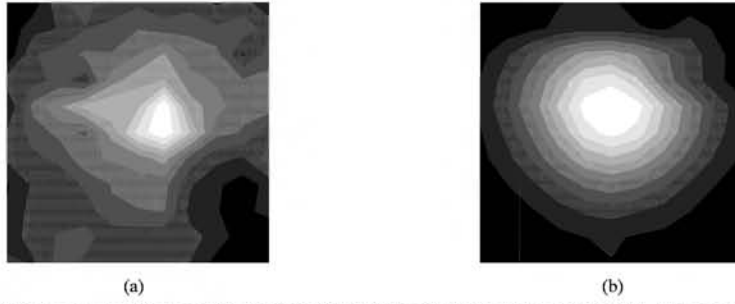

(a)                                          (b)

Figure 2: (a) A sample of spatial receptive field for a sLEC cell in our model. The lighter a region, the higher the cell's firing rate when the robot is in that region of the arena. (b) A typical place field in the CA3-CA1 layer of the model.

## 3 Hippocampal CA1-CA3 place field representation

When relying on visual data only, the state space representation encoded by place cells does not fulfill the Markov hypothesis [12]. Indeed, distinct areas of the environment may provide identical visual cues and lead to singularities in the view manifold (*sensory input aliasing*). We employ idiothetic signals along with visual information in order to remove such singularities and solve the hidden-state problem. An extra-hippocampal path integrator drives Gaussian-tuned neurons modeling self-motion information (sMEC in Fig. 1(a)). A fundamental contribution to build the sMEC idiothetic space representation comes from head-direction cells (projection B in Fig. 1(a)). As the robot moves, sMEC cell activity changes according to self-motion signals and to the current heading of the robot as estimated by the directional system. The firing activity $r_m$ of a cell $m \in$ sMEC is given by $r_m = \exp(-(\vec{s}_{dr} - \vec{s}_m)^2/2\sigma^2)$, where $\vec{s}_{dr}$ is the robot's current position estimated by dead-reckoning, $\vec{s}_m$ is the center of the receptive field of cell $m$, and $\sigma$ is the width of the Gaussian field.

Allothetic and idiothetic representations (i.e., sLEC and sMEC place field representations, respectively) converge onto CA3-CA1 regions to form a stable spatial representation (Fig. 1(a)). On the one hand, unreliable visual data are compensated for by means of path integration. On the other hand, reliable visual information can calibrate the path integrator system and maintain the dead-reckoning error bounded over time. Correlational learning is applied to combine visual cues and path integration over time. CA3-CA1 cells are recruited incrementally as exploration proceeds. For each new location, connections are established from all simultaneously active cells in sLEC and sMEC to newly recruited CA3-CA1 cells. Then, during the agent-environment interaction, Hebbian learning is applied to update the efficacy of the efferents from sLEC and sMEC to the hippocampus proper [11].

After learning, the CA3-CA1 space representation consists of a population of localized overlapping place fields (Fig. 2(b)) covering the two-dimensional workspace densely. Fig. 3(a) shows an example of distribution of CA3-CA1 place cells after learning. In this experiment, the robot, starting from an empty population, recruited about 1000 CA3-CA1 place cells.

In order to interpret the information represented by the ensemble CA3-CA1 pattern of activity, we employ *population vector coding* [13, 14]. Let $\vec{s}$ be the unknown robot's location, $r_i(\vec{s})$ the firing activity of a CA3-CA1 place cell $i$, and $\vec{s}_i$ the center of its place field. The population vector $\vec{p}$ is given by the center of mass of the network activity: $\vec{p} = \sum_i \vec{s}_i \, r_i(\vec{s}) / \sum_i r_i(\vec{s})$. The approximation $\vec{p} \approx \vec{s}$ is good for large neural populations covering the environment densely and uniformly [15]. In Fig. 3(a) the center of mass coding for the robot's location is represented by the black cross.

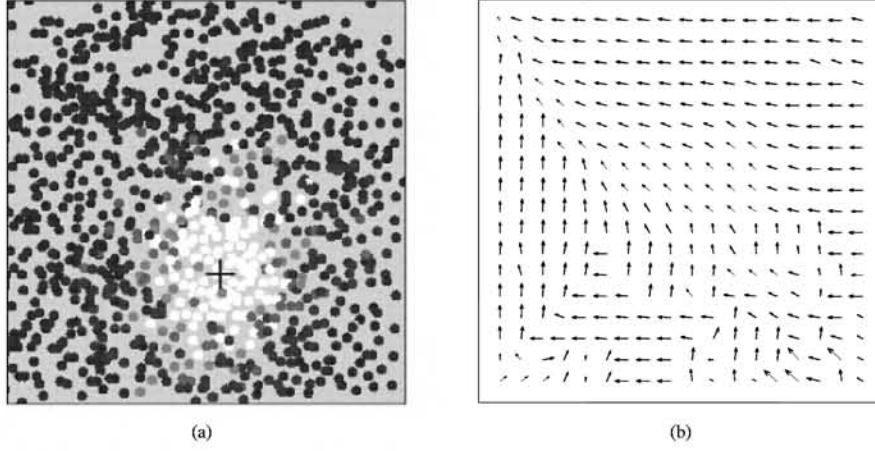

Figure 3: (a) The ensemble activity of approximately 1000 CA3-CA1 place cells created by the robot during spatial learning. Each dot is the center of a CA3-CA1 place cell. The lighter a cell, the higher its firing rate. The black cross is the center of mass of the ensemble activity. (b) Vector field representation of a navigational map learned after 5 trials. The target area (about 2.5 times the area occupied by the robot) is the upper-left corner of the arena.

## 4   Action learning: Goal-oriented navigation

The above spatial model enables the robot to localize itself within the environment. To support cognitive spatial behavior [1], the hippocampal circuit must also allow the robot to learn navigational maps autonomously. Our CA3-CA1 population provides an *incrementally learned coarse coding representation* suitable for applying reinforcement learning for continuous high-dimensional state spaces. Learning an action-value function over a continuous location space endows the system with spatial generalization capabilities.

We apply a $Q(\lambda)$ learning scheme [16] to build navigational maps [17, 18]. The overlapping localized CA3-CA1 place fields provide a natural set of basis functions that can be used to learn a parameterized form of the $Q(\lambda)$ function [19]. Note that we do not have to choose parameters like width and location of the basis functions. Rather, the basis functions are created automatically by unsupervised learning. Our representation also solves the problem of ambiguous input or partially hidden states [12], therefore the current state is fully known to the system and reinforcement learning can be applied in a straightforward manner.

Let $r_i$ denote the activation of a CA3-CA1 place cell $i$. Each state $\vec{s}$ is encoded by the ensemble place cell activity vector $\vec{r}(\vec{s}) = (r_1(\vec{s}), r_2(\vec{s}), \ldots, r_n(\vec{s}))$, where $n$ is the number of created place cells. The state-action value function $Q_w(\vec{s}, a)$ is of the form

$$Q_w(\vec{s}, a) = (\vec{w}^a)^T \, \vec{r}(\vec{s}) = \sum_{i=1}^{n} w_i^a \, r_i(\vec{s}) \tag{5}$$

where $\vec{s}, a$ is the state-action pair, and $\vec{w}^a = (w_1^a, \ldots, w_n^a)$ is an adjustable parameter vector. The learning task consists of updating the weight vector $\vec{w}^a$ to approximate the optimal function $Q_w^*(\vec{s}, a)$. The state-value prediction error is defined by

$$\delta_t = R_{t+1} + \gamma \, \max_a Q_t(\vec{s}_{t+1}, a) - Q_t(\vec{s}_t, a_t) \tag{6}$$

where $R_{t+1}$ is the immediate reward, and $0 \leq \gamma \leq 1$ is a constant discounting factor. At each time step the weight vector $\vec{w}^a$ changes according to

$$\vec{w}_{t+1}^a = \vec{w}_t^a + \alpha \delta_t \vec{e}_t \tag{7}$$

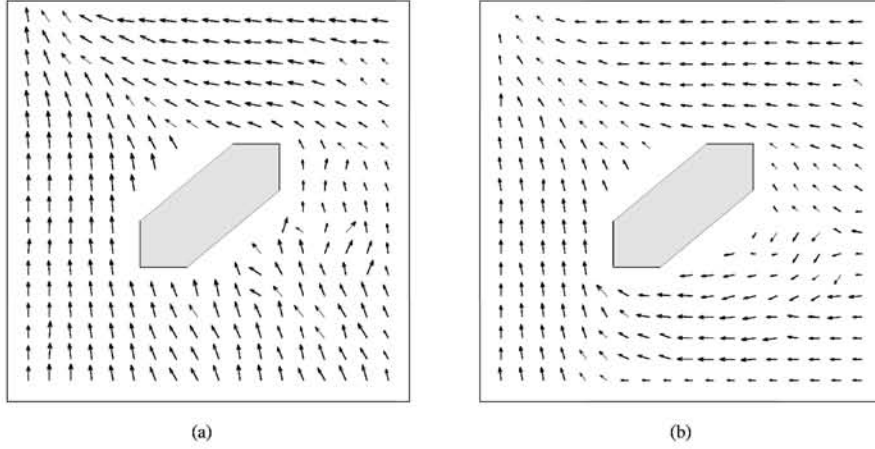

<center>(a)            (b)</center>

Figure 4: Two samples of learned navigational maps. The obstacle (dark grey object) is "transparent" with respect to vision, while it is detectable by the robot's infrared sensors. (a) The map learned by the robot after 20 training paths. (b) The map learned by the robot after 80 training trials.

where $0 \leq \alpha \leq 1$ is a constant learning rate parameter, and $\vec{e}_t$ is the eligibility trace vector. During learning, the exploitation-exploration trade-off is determined by an $\epsilon$-greedy policy, with $0 \leq \epsilon \leq 1$. As a consequence, at each step $t$ the agent might either behave greedily (*exploitation*) with probability $1 - \epsilon$, by selecting the best action $a_t^*$ with respect to the Q-value functions, $a_t^* = \mathrm{argmax}_a Q_t(\vec{s}_t, a)$, or resort to uniform random action selection (*exploration*) with probability equal to $\epsilon$.

The update of the eligibility trace depends on whether the robot selects an exploratory or an exploiting action. Specifically, the vector $\vec{e}_t$ changes according to (we start with $\vec{e}_0 = \vec{0}$)

$$\vec{e}_t = \vec{r}(\vec{s}_t) + \left\{ \begin{array}{ll} \gamma \lambda \vec{e}_{t-1} & \text{if } \textit{exploiting} \\ 0 & \text{if } \textit{exploring} \end{array} \right. \tag{8}$$

where $0 \leq \lambda \leq 1$ is a trace-decay parameter [19], and $\vec{r}(\vec{s}_t)$ is the CA3-CA1 vector activity.

Learning consists of a sequence of training paths starting at random positions and determined by the $\epsilon$-greedy policy. When the robot reaches the target, a new training path begins at a new random location. Fig. 3(b) shows an example of navigational map learned after 5 training trials. Fig. 4 shows some results obtained by adding an obstacle within the arena after the place field representation has been learned. Map of Fig. 4(a) has been learned after 20 training paths. It contains proper goal-oriented information, whereas it does not provide obstacle avoidance accurately[1]. Fig. 4(b) displays a navigational map learned by the robot after 80 training paths. Due to longer training, the map provides both appropriate goal-oriented and obstacle avoidance behavior. The vector field representations of Figs. 3(b) and 4 have been obtained by rastering uniformly over the environment. Many of sampled locations were not visited by the robot during training, which confirms the generalization capabilities of the method. That is, the robot was able to associate appropriate goal-oriented actions to never experienced spatial positions.

Reinforcement learning takes long training time when applied directly on high-dimensional input spaces [19]. We have shown that by means of an appropriate state space representation, based on localized overlapping place fields, the robot can learn goal-oriented behavior after only 5 training trials (without obstacles). This is similar to the escape platform learning time of rats in Morris water-maze [20].

**Acknowledgments**

Supported by the Swiss National Science Foundation, project nr. 21-49174.96.

## Footnotes

[1]Note that this does not really impair the robot's goal-oriented behavior, since obstacle avoidance is supported by a low-level reactive module driven by infrared sensors.

# References

[1] J. O'Keefe and L. Nadel. *The Hippocampus as a cognitive map.* Clarendon Press, Oxford, 1978.

[2] J.S. Taube, R.I. Muller, and J.B.Jr. Ranck. Head direction cells recorded from the postsubiculum in freely moving rats. I. Description and quantitative analysis. *Journal of Neuroscience*, 10:420–435, 1990.

[3] J. O'Keefe and N. Burgess. Geometric determinants of the place fields of hippocampal neurons. *Nature*, 381:425–428, 1996.

[4] M.O. Franz, B. Schölkopf, H.A. Mallot, and H.H. Bülthoff. Learning View Graphs for Robot Navigation. *Autonomous Robots*, 5:111–125, 1998.

[5] J.J. Knierim, H.S. Kudrimoti, and B.L. McNaughton. Place cells, head direction cells, and the learning of landmark stability. *Journal of Neuroscience*, 15:1648–1659, 1995.

[6] F. Smeraldi, J. Bigün, and W. Gerstner. On the role of dimensionality in face authentication. In *Proceedings of the Symposium of the Swedish Society for Automated Image Analysis, Halmstad (Sweden)*, pages 87–91. Halmstad University, Sweden, 2000.

[7] D. Gabor. Theory of communication. *Journal of the IEE*, 93:429–457, 1946.

[8] J.G. Daugman. Two-dimensional spectral analysis of cortical receptive field profiles. *Vision Research*, 20:847–856, 1980.

[9] F. Smeraldi, N. Capdevielle, and J. Bigün. Facial features detection by saccadic exploration of the Gabor decomposition and support vector machines. In *Proceedings of the 11th Scandinavian Conference on Image Analysis – SCIA 99, Kangerlussuaq, Greenland*, pages 39–44, 1999.

[10] A. Arleo and W. Gerstner. Modeling rodent head-direction cells and place cells for spatial learning in bio-mimetic robotics. In J.-A. Meyer, A. Berthoz, D. Floreano, H. Roitblat, and S.W. Wilson, editors, *From Animals to Animats VI*, pages 236–245, Cambridge MA, 2000. MIT Press.

[11] A. Arleo and W. Gerstner. Spatial cognition and neuro-mimetic navigation: A model of hippocampal place cell activity. *Biological Cybernetics, Special Issue on Navigation in Biological and Artificial Systems*, 83:287–299, 2000.

[12] R.A. McCallum. Hidden state and reinforcement learning with instance-based state identification. *IEEE Systems, Man, and Cybernetics*, 26(3):464–473, 1996.

[13] A.P. Georgopoulos, A. Schwartz, and R.E. Kettner. Neuronal population coding of movement direction. *Science*, 233:1416–1419, 1986.

[14] M.A. Wilson and B.L. McNaughton. Dynamics of the hippocampal ensemble code for space. *Science*, 261:1055–1058, 1993.

[15] E. Salinas and L.F. Abbott. Vector reconstruction from firing rates. *Journal of Computational Science*, 1:89–107, 1994.

[16] C.J.C.H. Watkins. *Learning from delayed rewards.* PhD thesis, University of Cambridge, England, 1989.

[17] P. Dayan. Navigating through temporal difference. In R.P. Lippmann, J.E. Moody, and D.S. Touretzky, editors, *Advances in Neural Information Processing Systems 3*, pages 464–470. Morgan Kaufmann, San Mateo, CA, 1991.

[18] D.J. Foster, R.G.M. Morris, and P. Dayan. A model of hippocampally dependent navigation, using the temporal difference learning rule. *Hippocampus*, 10(1):1–16, 2000.

[19] R.S. Sutton and A.G. Barto. *Reinforcement learning, an introduction.* MIT Press-Bradford Books, Cambridge, Massachusetts, 1998.

[20] R.G.M. Morris, P. Garrud, J.N.P. Rawlins, and J. O'Keefe. Place navigation impaired in rats with hippocampal lesions. *Nature*, 297:681–683, 1982.
